# Learning to Explore and Exploit in POMDPs

**Chenghui Cai, Xuejun Liao, and Lawrence Carin**
Department of Electrical and Computer Engineering
Duke University
Durham, NC 27708-0291, USA

## Abstract

A fundamental objective in reinforcement learning is the maintenance of a proper balance between exploration and exploitation. This problem becomes more challenging when the agent can only partially observe the states of its environment. In this paper we propose a dual-policy method for jointly learning the agent behavior and the balance between exploration exploitation, in partially observable environments. The method subsumes traditional exploration, in which the agent takes actions to gather information about the environment, and active learning, in which the agent queries an oracle for optimal actions (with an associated cost for employing the oracle). The form of the employed exploration is dictated by the specific problem. Theoretical guarantees are provided concerning the optimality of the balancing of exploration and exploitation. The effectiveness of the method is demonstrated by experimental results on benchmark problems.

## 1 Introduction

A fundamental challenge facing reinforcement learning (RL) algorithms is to maintain a proper balance between exploration and exploitation. The policy designed based on previous experiences is by construction constrained, and may not be optimal as a result of inexperience. Therefore, it is desirable to take actions with the goal of enhancing experience. Although these actions may not necessarily yield optimal *near-term* reward toward the ultimate goal, they could, over a long horizon, yield improved *long-term* reward. The fundamental challenge is to achieve an optimal balance between exploration and exploitation; the former is performed with the goal of enhancing experience and preventing premature convergence to suboptimal behavior, and the latter is performed with the goal of employing available experience to define perceived optimal actions.

For a Markov decision process (MDP), the problem of balancing exploration and exploitation has been addressed successfully by the $E^3$ [4, 5] and R-max [2] algorithms. Many important applications, however, have environments whose states are not completely observed, leading to partially observable MDPs (POMDPs). Reinforcement learning in POMDPs is challenging, particularly in the context of balancing exploration and exploitation. Recent work targeted on solving the exploration vs. exploitation problem is based on an augmented POMDP, with a product state space over the environment states and the unknown POMDP parameters [9]. This, however, entails solving a complicated planning problem, which has a state space that grows exponentially with the number of unknown parameters, making the problem quickly intractable in practice. To mitigate this complexity, active learning methods have been proposed for POMDPs, which borrow similar ideas from supervised learning, and apply them to selectively query an oracle (domain expert) for the optimal action [3]. Active learning has found success in many collaborative human-machine tasks where expert advice is available.

In this paper we propose a dual-policy approach to balance exploration and exploitation in POMDPs, by simultaneously learning two policies with partially shared internal structure. The first policy, termed the *primary policy*, defines actions based on previous experience; the second policy, termed

the *auxiliary policy*, is a meta-level policy maintaining a proper balance between exploration and exploitation. We employ the regionalized policy representation (RPR) [6] to parameterize both policies, and perform Bayesian learning to update the policy posteriors. The approach applies in either of two cases: (*i*) the agent explores by randomly taking the actions that have been insufficiently tried before (traditional exploration), or (*ii*) the agent explores by querying an oracle for the optimal action (active learning). In the latter case, the agent is assessed a query cost from the oracle, in addition to the reward received from the environment. Either (*i*) or (*ii*) is employed as an exploration vehicle, depending upon the application.

The dual-policy approach possesses interesting convergence properties, similar to those of E$^3$ [5] and Rmax [2]. However, our approach assumes the environment is a POMDP while E$^3$ and Rmax both assume an MDP environment. Another distinction is that our approach learns the agent policy directly from episodes, without estimating the POMDP model. This is in contrast to E$^3$ and Rmax (both learn MDP models) and the active-learning method in [3] (which learns POMDP models).

## 2 Regionalized Policy Representation

We first provide a brief review of the regionalized policy representation, which is used to parameterize the primary policy and the auxiliary policy as discussed above. The material in this section is taken from [6], with the proofs omitted here.

**Definition 2.1** *A regionalized policy representation is a tuple $(\mathcal{A}, \mathcal{O}, \mathcal{Z}, W, \mu, \pi)$. The $\mathcal{A}$ and $\mathcal{O}$ are respectively a finite set of actions and observations. The $\mathcal{Z}$ is a finite set of belief regions. The $W$ is the belief-region transition function with $W(z, a, o', z')$ denoting the probability of transiting from $z$ to $z'$ when taking action $a$ in $z$ results in observing $o'$. The $\mu$ is the initial distribution of belief regions with $\mu(z)$ denoting the probability of initially being in $z$. The $\pi$ are the region-dependent stochastic policies with $\pi(z, a)$ denoting the probability of taking action $a$ in $z$.*

We denote $\mathcal{A} = \{1, 2, \ldots, |\mathcal{A}|\}$, where $|\mathcal{A}|$ is the cardinality of $\mathcal{A}$. Similarly, $\mathcal{O} = \{1, 2, \ldots, |\mathcal{O}|\}$ and $\mathcal{Z} = \{1, 2, \ldots, |\mathcal{Z}|\}$. We abbreviate $(a_0, a_1, \ldots, a_T)$ as $a_{0:T}$ and similarly, $(o_1, o_2, \ldots, a_T)$ as $o_{1:T}$ and $(z_0, z_1, \ldots, z_T)$ as $z_{0:T}$, where the subscripts indexes discrete time steps. The history $h_t = \{a_{0:t-1}, o_{1:t}\}$ is defined as a sequence of actions performed and observations received up to $t$. Let $\Theta = \{\pi, \mu, W\}$ denote the RPR parameters. Given $h_t$, the RPR yields a joint probability distribution of $z_{0:t}$ and $a_{0:t}$ as follows

$$p(a_{0:t}, z_{0:t} | o_{1:t}, \Theta) = \mu(z_0)\pi(z_0, a_0)\prod_{\tau=1}^{t} W(z_{\tau-1}, a_{\tau-1}, o_\tau, z_\tau)\pi(z_\tau, a_\tau) \tag{1}$$

By marginalizing $z_{0:t}$ out in (1), we obtain $p(a_{0:t} | o_{1:t}, \Theta)$. Furthermore, the history-dependent distribution of action choices is obtained as follows:

$$p(a_\tau | h_\tau, \Theta) = p(a_{0:\tau} | o_{1:\tau}, \Theta)[p(a_{0:\tau-1} | o_{1:\tau-1}, \Theta)]^{-1}$$

which gives a stochastic policy for choosing the action $a_\tau$. The action choice depends solely on the historical actions and observations, with the unobservable belief regions marginalized out.

### 2.1 Learning Criterion

Bayesian learning of the RPR is based on the experiences collected from the agent-environment interaction. Assuming the interaction is episodic, i.e., it breaks into subsequences called episodes [10], we represent the experiences by a set of episodes.

**Definition 2.2** *An episode is a sequence of agent-environment interactions terminated in an absorbing state that transits to itself with zero reward. An episode is denoted by $(a_0^k r_0^k o_1^k a_1^k r_1^k \cdots o_{T_k}^k a_{T_k}^k r_{T_k}^k)$, where the subscripts are discrete times, $k$ indexes the episodes, and $o$, $a$, and $r$ are respectively observations, actions, and immediate rewards.*

**Definition 2.3** *(The RPR Optimality Criterion) Let $\mathcal{D}^{(K)} = \{(a_0^k r_0^k o_1^k a_1^k r_1^k \cdots o_{T_k}^k a_{T_k}^k r_{T_k}^k)\}_{k=1}^{K}$ be a set of episodes obtained by an agent interacting with the environment by following policy $\Pi$ to select actions, where $\Pi$ is an arbitrary stochastic policy with action-selecting distributions $p^\Pi(a_t | h_t) > 0$, $\forall$ action $a_t$, $\forall$ history $h_t$. The RPR optimality criterion is defined as*

$$\widehat{V}(\mathcal{D}^{(K)}; \Theta) \overset{def.}{=} \frac{1}{K}\sum_{k=1}^{K}\sum_{t=0}^{T_k}\gamma^t r_t^k \frac{\prod_{\tau=0}^{t} p(a_\tau^k | h_\tau^k, \Theta)}{\prod_{\tau=0}^{t} p^\Pi(a_\tau^k | h_\tau^k)} \tag{2}$$

where $h_t^k = a_0^k o_1^k a_1^k \cdots o_t^k$ is the history of actions and observations up to time $t$ in the $k$-th episode, $0 < \gamma < 1$ is the discount, and $\Theta$ denotes the RPR parameters.

Throughout the paper, we call $\widehat{V}(\mathcal{D}^{(K)}; \Theta)$ the empirical value function of $\Theta$. It is proven in [6] that $\lim_{K \to \infty} \widehat{V}(\mathcal{D}^{(K)}; \Theta)$ is the expected sum of discounted rewards by following the RPR policy parameterized by $\Theta$ for an infinite number of steps. Therefore, the RPR resulting from maximization of $\widehat{V}(\mathcal{D}^{(K)}; \Theta)$ approaches the optimal as $K$ is large (assuming $|Z|$ is appropriate). In the Bayesian setting discussed below, we use a noninformative prior for $\Theta$, leading to a posterior of $\Theta$ peaked at the optimal RPR, therefore the agent is guaranteed to sample the optimal or a near-optimal policy with overwhelming probability.

## 2.2 Bayesian Learning

Let $G_0(\Theta)$ represent the prior distribution of the RPR parameters. We define the posterior of $\Theta$ as

$$p(\Theta|\mathcal{D}^{(K)}, G_0) \overset{def.}{=} \widehat{V}(\mathcal{D}^{(K)}; \Theta) G_0(\Theta) [\widehat{V}(\mathcal{D}^{(K)})]^{-1} \tag{3}$$

where $\widehat{V}(\mathcal{D}^{(K)}) = \int \widehat{V}(\mathcal{D}^{(K)}; \Theta) G_0(\Theta) d\Theta$ is the marginal empirical value. Note that $\widehat{V}(\mathcal{D}^{(K)}; \Theta)$ is an empirical value function, thus (3) is a non-standard use of Bayes rule. However, (3) indeed gives a distribution whose shape incorporates both the prior and the empirical information.

Since each term in $\widehat{V}(\mathcal{D}^{(K)}; \Theta)$ is a product of multinomial distributions, it is natural to choose the prior as a product of Dirichlet distributions,

$$G_0(\Theta) = p(\mu|\upsilon) p(\pi|\rho) p(W|\omega) \tag{4}$$

where $p(\mu|\upsilon) = \mathrm{Dir}(\mu(1), \cdots, \mu(|\mathcal{Z}|)|\upsilon)$, $p(\pi|\rho) = \prod_{i=1}^{|\mathcal{Z}|} \mathrm{Dir}(\pi(i, 1), \cdots, \pi(i, |\mathcal{A}|)|\rho_i)$, $p(W|\omega) = \prod_{a=1}^{|\mathcal{A}|} \prod_{o=1}^{|\mathcal{O}|} \prod_{i=1}^{|\mathcal{Z}|} \mathrm{Dir}(W(i, a, o, 1), \cdots, W(i, a, o, |\mathcal{Z}|)|\omega_{i,a,o})$; $\rho_i = \{\rho_{i,m}\}_{m=1}^{|\mathcal{A}|}$, $\upsilon = \{\upsilon_i\}_{i=1}^{|\mathcal{Z}|}$, and $\omega_{i,a,o} = \{\omega_{i,a,o,j}\}_{j=1}^{|\mathcal{Z}|}$ are hyper-parameters. With the prior thus chosen, the posterior in (3) is a large mixture of Dirichlet products, and therefore posterior analysis by Gibbs sampling is inefficient. To overcome this, we employ the variational Bayesian technique [1] to obtain a variational posterior by maximizing a lower bound to $\ln \int \widehat{V}(\mathcal{D}^{(K)}; \Theta) G_0(\Theta) d\Theta$,

$$\mathrm{LB}(\{q_t^k\}, g(\Theta)) = \ln \int \widehat{V}(\mathcal{D}^{(K)}; \Theta) G_0(\Theta) d\Theta - \mathrm{KL}(\{q_t^k(z_{0:t}^k) g(\Theta)\} || \{\nu_t^k p(z_{0:t}^k, \Theta|a_{0:t}^k, o_{1:t}^k)\})$$

where $\{q_t^k\}, g(\Theta)$ are variational distributions satisfying $q_t^k(z_{0:t}^k) \geq 1$, $g(\Theta) \geq 1$, $\int g(\Theta) d\Theta = 1$, and $\frac{1}{K} \sum_{k=1}^{K} \sum_{t=1}^{T_k} \sum_{z_0^k, \cdots, z_t^k = 1}^{|\mathcal{Z}|} q_t^k(z_{0:t}^k) = 1$; $\nu_t^k = \frac{\gamma^t r_t^k p(a_{0:t}^k|o_{1:t}^k)}{\prod_{\tau=0}^{t} p^{\Pi}(a_\tau^k|h_\tau^k) \widehat{V}(\mathcal{D}^{(K)})}$ and $\mathrm{KL}(q\|p)$ denotes the Kullback-Leibler (KL) distance between probability measure $q$ and $p$.

The factorized form $\{q_t(z_{0:t}) g(\Theta)\}$ represents an approximation of the weighted joint posterior of $\Theta$ and $z$'s when the lower bound reaches the maximum, and the corresponding $g(\Theta)$ is called the variational approximate posterior of $\Theta$. The lower bound maximization is accomplished by solving $\{q_t(z_{0:t})\}$ and $g(\Theta)$ alternately, keeping one fixed while solving for the other. The solutions are summarized in Theorem 2.4; the proof is in [6].

**Theorem 2.4** *Given the initialization $\widehat{\rho} = \rho$, $\widehat{\upsilon} = \upsilon$, $\widehat{\omega} = \omega$, iterative application of the following updates produces a sequence of monotonically increasing lower bounds LB($\{q_t^k\}, g(\Theta)$), which converges to a maxima. The update of $\{q_t^k\}$ is*

$$q_z^k(z_{0:t}^k) = \sigma_t^k p(z_{0:t}^k|a_{0:t}^k, o_{1:t}^k, \widetilde{\Theta})$$

*where $\widetilde{\Theta} = \{\widetilde{\pi}, \widetilde{\mu}, \widetilde{W}\}$ is a set of under-normalized probability mass functions, with $\widetilde{\pi}(i, m) = e^{\psi(\widehat{\rho}_{i,m}) - \psi(\sum_{m=1}^{|\mathcal{A}|} \widehat{\rho}_{i,m})}$, $\widetilde{\mu}(i) = e^{\psi(\widehat{\upsilon}_i) - \psi(\sum_{i=1}^{|\mathcal{Z}|} \widehat{\upsilon}_i)}$, and $\widetilde{W}(i, a, o, j) = e^{\psi(\widehat{\omega}_{i,a,o,j}) - \psi(\sum_{j=1}^{|\mathcal{A}|} \widehat{\omega}_{i,a,o,j})}$, and $\psi$ is the digamma function. The $g(\Theta)$ has the same form as the prior $G_0$ in (4), except that the hyper-parameter are updated as*

$$\widehat{\upsilon}_i = \upsilon_i + \sum_{k=1}^{K} \sum_{t=0}^{T_k} \sigma_t^k \phi_{t,0}^k(i)$$

$$\begin{aligned}
\widehat{\rho}_{i,a} &= \rho_{i,a} + \sum_{k=1}^{K}\sum_{t=0}^{T_k}\sum_{\tau=0}^{t}\sigma_t^k\phi_{t,\tau}^k(i)\delta(a_\tau^k,a) \\
\widehat{\omega}_{i,a,o,j} &= \omega_{i,a,o,j} + \sum_{k=1}^{K}\sum_{t=0}^{T_k}\sum_{\tau=1}^{t}\sigma_t^k\xi_{t,\tau-1}^k(i,j)\delta(a_{\tau-1}^k,a)\delta(o_\tau^k,o)
\end{aligned}$$

*where* $\xi_{t,\tau}^k(i,j) = p(z_\tau^k = i, z_{\tau+1}^k = j|a_{0:t}^k, o_{1:t}^k, \widetilde{\Theta})$, $\phi_{t,\tau}^k(i) = p(z_\tau^k = i|a_{0:t}^k, o_{1:t}^k, \widetilde{\Theta})$, *and*

$$\sigma_t^k = \left[\gamma^t r_t^k p(a_{0:t}^k|o_{1:t}^k, \widetilde{\Theta})\right]\left[\prod_{\tau=0}^{t}p^\Pi(a_\tau^k|h_\tau^k)\widehat{V}(\mathcal{D}^{(K)}|\widetilde{\Theta})\right]^{-1} \tag{5}$$

## 3 Dual-RPR: Joint Policy for the Agent Behavior and the Trade-Off Between Exploration and Exploitation

Assume that the agent uses the RPR described in Section 2 to govern its behavior in the unknown POMDP environment (the primary policy). Bayesian learning employs the empirical value function $\widehat{V}(\mathcal{D}^{(K)}; \Theta)$ in (2) in place of a likelihood function, to obtain the posterior of the RPR parameters $\Theta$. The episodes $\mathcal{D}^{(K)}$ may be obtained from the environment by following an arbitrary stochastic policy $\Pi$ with $p^\Pi(a|h) > 0$, $\forall a$, $\forall h$. Although any such $\Pi$ guarantees optimality of the resulting RPR, the choice of $\Pi$ affects the convergence speed. A good choice of $\Pi$ avoids episodes that do not bring new information to improve the RPR, and thus the agent does not have to see all possible episodes before the RPR becomes optimal.

In batch learning, all episodes are collected before the learning begins, and thus $\Pi$ is pre-chosen and does not change during the learning [6]. In online learning, however, the episodes are collected during the learning, and the RPR is updated upon completion of each episode. Therefore there is a chance to exploit the RPR to avoid repeated learning in the same part of the environment. The agent should recognize belief regions it is familiar with, and exploit the existing RPR policy there; in belief regions inferred as new, the agent should explore. This balance between exploration and exploitation is performed with the goal of accumulating a large long-run reward.

We consider online learning of the RPR (as the primary policy) and choose $\Pi$ as a mixture of two policies: one is the current RPR $\Theta$ (exploitation) and the other is an exploration policy $\Pi_e$. This gives the action-choosing probability $p^\Pi(a|h) = p(y = 0|h)p(a|h, \Theta, y = 0) + p(y = 1|h)p(a|h, \Pi_e, y = 1)$, where $y = 0$ ($y = 1$) indicates exploitation (exploration). The problem of choosing good $\Pi$ then reduces to a proper balance between exploitation and exploration: the agent should exploit $\Theta$ when doing so is highly rewarding, while following $\Pi_e$ to enhance experience and improve $\Theta$.

An *auxiliary RPR* is employed to represent the policy for balancing exploration and exploitation, i.e., the history-dependent distribution $p(y|h)$. The auxiliary RPR shares the parameters $\{\mu, W\}$ with the primary RPR, but with $\pi = \{\pi(z, a) : a \in \mathcal{A}, z \in \mathcal{Z}\}$ replaced by $\lambda = \{\lambda(z, y) : y = 0 \text{ or } 1, z \in \mathcal{Z}\}$, where $\lambda(z, y)$ is the probability of choosing exploitation ($y = 0$) or exploration ($y = 1$) in belief region $z$. Let $\lambda$ have the prior

$$p(\lambda|u) = \prod_{i=1}^{|\mathcal{Z}|}\text{Beta}\Big(\lambda(i,0), \lambda(i,1)\Big|u_0, u_1\Big). \tag{6}$$

In order to encourage exploration when the agent has little experience, we choose $u_0 = 1$ and $u_1 > 1$ so that, at the beginning of learning, the auxiliary RPR always suggests exploration. As the agent accumulates episodes of experience, it comes to know a certain part of the environment in which the episodes have been collected. This knowledge is reflected in the auxiliary RPR, which, along with the primary RPR, is updated upon completion of each new episode.

Since the environment is a POMDP, the agent's knowledge should be represented in the space of belief states. However, the agent cannot directly access the belief states, because computation of belief states requires knowing the true POMDP model, which is not available. Fortunately, the RPR formulation provides a compact representation of $\mathcal{H} = \{h\}$, the space of histories, where each history $h$ corresponds to a belief state in the POMDP. Within the RPR formulation, $\mathcal{H}$ is represented internally as the set of distributions over belief regions $z \in \mathcal{Z}$, which allows the agent to access $\mathcal{H}$ based on a subset of samples from $\mathcal{H}$. Let $\mathcal{H}_{\text{known}}$ be the part of $\mathcal{H}$ that has become known to the agent, i.e., the primary RPR is optimal in $\mathcal{H}_{\text{known}}$ and thus the agent should begin to exploit upon entering $\mathcal{H}_{\text{known}}$. As will be clear below, $\mathcal{H}_{\text{known}}$ can be identified by $\mathcal{H}_{\text{known}} = \{h : p(y = 0|h, \Theta, \lambda) \approx 1\}$, if the posterior of $\lambda$ is updated by

$$\widehat{u}_{i,0} = u_0 + \sum_{k=1}^{K}\sum_{t=0}^{T_k}\sum_{\tau=0}^{t}\sigma_t^k\phi_{t,\tau}^k(i), \tag{7}$$

$$\widehat{u}_{i,1} = \max\Big(\eta, u_1 - \sum_{k=1}^{K}\sum_{t=0}^{T_k}\sum_{\tau=0}^{t}y_t^k\gamma^t c\,\phi_{t,\tau}^k(i)\Big), \tag{8}$$

where $\eta$ is a small positive number, and $\sigma_t^k$ is the same in (5) except that $r_t^k$ is replaced by $m_t^k$, the meta-reward received at $t$ in episode $k$. We have $m_t^k = r_{\text{meta}}$ if the goal is reached at time $t$ in episode $k$, and $m_t^k = 0$ otherwise, where $r_{\text{meta}} > 0$ is a constant. When $\Pi_e$ is provided by an oracle (active learning), a query cost $c > 0$ is taken into account in (8), by subtracting $c$ from $u_1$. Thus, the probability of exploration is reduced each time the agent makes a query to the oracle (i.e., $y_t^k = 1$). After a certain number of queries, $\hat{u}_{i,1}$ becomes the small positive number $\eta$ (it never becomes zero due to the max operator), at which point the agent stops querying in belief region $z = i$.

In (7) and (8), exploitation always receives a "credit", while exploration never receives credit (exploration is actually discredited when $\Pi_e$ is an oracle). This update makes sure that the chance of exploitation monotonically increases as the episodes accumulate. Exploration receives no credit because it has been pre-assigned a credit ($u_1$) in the prior, and the chance of exploration should monotonically decrease with the accumulation of episodes. The parameter $u_1$ represents the agent's prior for the amount of needed exploration. When $c > 0$, $u_1$ is discredited by the cost and the agent needs a larger $u_1$ (than when $c = 0$) to obtain the same amount of exploration. The fact that the amount of exploration monotonically increases with $u_1$ implies that, one can always find a large enough $u_1$ to ensure that the primary RPR is optimal in $\mathcal{H}_{\text{known}} = \{h : p(y = 0|h, \Theta, \lambda) \approx 1\}$. However, an unnecessarily large $u_1$ makes the agent over-explore and leads to slow convergence. Let $u_1^{\text{min}}$ denote the minimum $u_1$ that ensures optimality in $\mathcal{H}_{\text{known}}$. We assume $u_1^{\text{min}}$ exists in the analysis below. The possible range of $u_1^{\text{min}}$ is examined in the experiments.

## 4 Optimality and Convergence Analysis

Let $M$ be the true POMDP model. We first introduce an equivalent expression for the empirical value function in (2),

$$\widehat{V}(\mathcal{E}_T^{(K)}; \Theta) = \sum_{\mathcal{E}_T^{(K)}} \sum_{t=0}^{T} \gamma^t r_t p(a_{0:t}, o_{1:t}, r_t | y_{0:t} = 0, \Theta, M), \qquad (9)$$

where the first summation is over all elements in $\mathcal{E}_T^{(K)} \subseteq \mathcal{E}_T$, and $\mathcal{E}_T = \{(a_{0:T}, o_{1:T}, r_{0:T}) : a_t \in \mathcal{A}, o_t \in \mathcal{O}, t = 0, 1, \cdots, T\}$ is the complete set of episodes of length $T$ in the POMDP, with no repeated elements. The condition $y_{0:t} = 0$, which is an an abbreviation for $y_\tau = 0 \, \forall \, \tau = 0, 1, \cdots, t$, indicates that the agent always follows the RPR ($\Theta$) here. Note $\widehat{V}(\mathcal{E}_T^{(K)}; \Theta)$ is the empirical value function of $\Theta$ defined on $\mathcal{E}_T^{(K)}$, as is $\widehat{V}(\mathcal{D}^{(K)}; \Theta)$ on $\mathcal{D}^{(K)}$. When $T = \infty$ [1], the two are identical up to a difference in acquiring the episodes: $\mathcal{E}_T^{(K)}$ is a simple enumeration of distinct episodes while $\mathcal{D}^{(K)}$ may contain identical episodes. The multiplicity of an episode in $\mathcal{D}^{(K)}$ results from the sampling process (by following a policy to interact with the environment). Note that the empirical value function defined using $\mathcal{E}_T^{(K)}$ is interesting only for theoretical analysis, because the evaluation requires knowing the true POMDP model, not available in practice. We define the optimistic value function

$$\widehat{V}_f(\mathcal{E}_T^{(K)}; \Theta, \lambda, \Pi_e) = \sum_{\mathcal{E}_T^{(K)}} \sum_{t=0}^{T} \gamma^t \sum_{y_0, \cdots, y_t=0}^{1} \left(r_t + (R_{\max} - r_t) \vee_{\tau=0}^t y_\tau\right) p(a_{0:t}, o_{1:t}, r_t, y_{0:t} | \Theta, \lambda, M, \Pi_e) \quad (10)$$

where $\vee_{\tau=0}^t y_\tau$ indicates that the agent receives $r_t$ if and only if $y_\tau = 0$ at all time steps $\tau = 1, 2, \cdots, t$; otherwise, it receives $R_{\max}$ at $t$, which is an upper bound of the rewards in the environment. Similarly we can define $\widehat{V}(\mathcal{D}^{(K)}; \Theta, \lambda, \Pi_e)$, the equivalent expression for $\widehat{V}_f(\mathcal{E}_T^{(K)}; \Theta, \lambda, \Pi_e)$. The following lemma is proven in the Appendix.

**Lemma 4.1** *Let* $\widehat{V}(\mathcal{E}_T^{(K)}; \Theta)$, $\widehat{V}_f(\mathcal{E}_T^{(K)}; \Theta, \lambda, \Pi_e)$, *and* $R_{\max}$ *be defined as above. Let* $P_{\text{exlpore}}(\mathcal{E}_T^{(K)}, \Theta, \lambda, \Pi_e)$ *be the probability of executing the exploration policy* $\Pi_e$ *at least once in some episode in* $\mathcal{E}_T^{(K)}$, *under the auxiliary RPR* $(\Theta, \lambda)$ *and the exploration policy* $\Pi_e$. *Then*

$$P_{\text{exlpore}}(\mathcal{E}_T^{(K)}, \Theta, \lambda, \Pi_e) \geq \frac{1 - \gamma}{R_{\max}} |\widehat{V}(\mathcal{E}_T^{(K)}; \Theta) - \widehat{V}_f(\mathcal{E}_T^{(K)}; \Theta, \lambda, \Pi_e)|.$$

**Proposition 4.2** *Let $\Theta$ be the optimal RPR on $\mathcal{E}_\infty^{(K)}$ and $\Theta^*$ be the optimal RPR in the complete POMDP environment. Let the auxiliary RPR hyper-parameters ($\lambda$) be updated according to (7) and (8), with $u_1 \geq u_1^{\min}$. Let $\Pi_e$ be the exploration policy and $\epsilon \geq 0$. Then either (a) $\widehat{V}(\mathcal{E}_\infty; \Theta) \geq \widehat{V}(\mathcal{E}_\infty; \Theta^*) - \epsilon$, or (b) the probability that the auxiliary RPR suggests executing $\Pi_e$ in some episode unseen in $\mathcal{E}_\infty^{(K)}$ is at least $\frac{\epsilon(1-\gamma)}{R_{\max}}$.*

**Proof:** It is sufficient to show that if (a) does not hold, then (b) must hold. Let us assume $\widehat{V}(\mathcal{E}_\infty; \Theta) < \widehat{V}(\mathcal{E}_\infty; \Theta^*) - \epsilon$. Because $\Theta$ is optimal in $\mathcal{E}_\infty^{(K)}$, $\widehat{V}(\mathcal{E}_\infty^{(K)}; \Theta) \geq \widehat{V}(\mathcal{E}_\infty^{(K)}; \Theta^*)$, which implies $\widehat{V}(\mathcal{E}_\infty^{(\backslash K)}; \Theta) < \widehat{V}(\mathcal{E}_\infty^{(\backslash K)}; \Theta^*) - \epsilon$. where $\mathcal{E}_\infty^{(\backslash K)} = \mathcal{E}_\infty \setminus \mathcal{E}_\infty^{(K)}$. We show below that $\widehat{V}_f(\mathcal{E}_\infty^{(\backslash K)}; \Theta, \lambda, \Pi_e) \geq \widehat{V}(\mathcal{E}_\infty^{(\backslash K)}; \Theta^*)$ which, together with Lemma 4.1, implies

$$
\begin{aligned}
P_{\text{exlpore}}(\mathcal{E}_\infty^{(\backslash K)}, \Theta, \lambda, \Pi_e) &\geq \frac{1-\gamma}{R_{\max}} \left[ \widehat{V}_f(\mathcal{E}_\infty^{(\backslash K)}; \Theta, \lambda, \Pi_e) - \widehat{V}(\mathcal{E}_\infty^{(\backslash K)}; \Theta) \right] \\
&\geq \frac{1-\gamma}{R_{\max}} \left[ \widehat{V}(\mathcal{E}_\infty^{(\backslash K)}; \Theta^*) - \widehat{V}(\mathcal{E}_\infty^{(\backslash K)}; \Theta) \right] \geq \frac{\epsilon(1-\gamma)}{R_{\max}}
\end{aligned}
$$

We now show $\widehat{V}_f(\mathcal{E}_\infty^{(\backslash K)}; \Theta, \lambda, \Pi_e) \geq \widehat{V}(\mathcal{E}_\infty^{(\backslash K)}; \Theta^*)$. By construction, $\widehat{V}_f(\mathcal{E}_\infty^{(\backslash K)}; \Theta, \lambda, \Pi_e)$ is an optimistic value function, in which the agent receives $R_{\max}$ at any time $t$ unless if $y_\tau = 0$ at $\tau = 0, 1, \cdots, t$. However, $y_\tau = 0$ at $\tau = 0, 1, \cdots, t$ implies that $\{h_\tau : \tau = 0, 1, \cdots, t\} \subset \mathcal{H}_{\text{known}}$, By the premise, $\lambda$ is updated according to (7) and (8) and $u_1 \geq u_1^{\min}$, therefore $\Theta$ is optimal in $\mathcal{H}_{\text{known}}$ (see the discussions following (7) and (8)), which implies $\Theta$ is optimal in $\{h_\tau : \tau = 0, 1, \cdots, t\}$. Thus, the inequality holds. Q.E.D.

Proposition 4.2 shows that whenever the primary RPR achieves less accumulative reward than the optimal RPR by $\epsilon$, the auxiliary RPR suggests exploration with a probability exceeding $\epsilon(1-\gamma)R_{\max}^{-1}$. Conversely, whenever the auxiliary RPR suggests exploration with a probability smaller than $\epsilon(1-\gamma)R_{\max}^{-1}$, the primary RPR achieves $\epsilon$-near optimality. This ensures that the agent is either receiving sufficient rewards or it is performing sufficient exploration.

## 5 Experimental Results

Our experiments are based on Shuttle, a benchmark POMDP problem [7], with the following setup. The primary policy is a RPR with $|\mathcal{Z}| = 10$ and a prior in (4), with all hyper-parameters initially set to one (which makes the initial prior non-informative). The auxiliary policy is a RPR sharing $\{\mu, W\}$ with the primary RPR and having a prior for $\lambda$ as in (6). The prior of $\lambda$ is initially biased towards exploration by using $u_0 = 1$ and $u_1 > 1$. We consider various values of $u_1$ to examine the different effects. The agent performs online learning: upon termination of each new episode, the primary and auxiliary RPR posteriors are updated by using the previous posteriors as the current priors. The primary RPR update follows Theorem 2.4 with $K = 1$ while the auxiliary RPR update follows (7) and (8) for $\lambda$ (it shares the same update with the primary RPR for $\mu$ and $W$). We perform 100 independent Monte Carlo runs. In each run, the agent starts learning from a random position in the environment and stops learning when $K_{\text{total}}$ episodes are completed. We compare various methods that the agent uses to balance exploration and exploitation: (*i*) following the auxiliary RPR, with various values of $u_1$, to adaptively switch between exploration and exploitation; (*ii*) randomly switching between exploration and exploitation with a fixed exploration rate $P_{\text{explore}}$ (various values of $P_{\text{explore}}$ are examined). When performing exploitation, the agent follows the current primary RPR (using the $\Theta$ that maximizes the posterior); when performing exploration, it follows an exploration policy $\Pi_e$. We consider two types of $\Pi_e$: (*i*) taking random actions and (*ii*) following the policy obtained by solving the *true* POMDP using PBVI [8] with 2000 belief samples. In either case, $r_{\text{meta}} = 1$ and $\eta = 0.001$. In case (*ii*), the PBVI policy is the oracle and incurs a query cost $c$.

We report: (*i*) the sum of discounted rewards accrued within each episode during learning; these rewards result from both exploitation and exploration. (*ii*) the quality of the primary RPR upon termination of each learning episode, represented by the sum of discounted rewards averaged over 251 episodes of following the primary RPR (using the standard testing procedure for Shuttle: each episode is terminated when either the goal is reached or a maximum of 251 steps is taken); these rewards result from exploitation alone. (*iii*) the exploration rate $P_{\text{explore}}$ in each learning episode, which is the number of time steps at which exploration is performed divided by the total time steps in

a given episode. In order to examine the optimality, the rewards in (*i*)-(*ii*) has the corresponding optimal rewards subtracted, where the optimal rewards are obtained by following the PBVI policy; the difference are reported, with zero difference indicating optimality and minus difference indicating sub-optimality. All results are averaged over the 100 Monte Carlo runs. The results are summarized in Figure 1 when $\Pi_e$ takes random actions and in Figure 2 when $\Pi_e$ is an oracle (the PBVI policy).

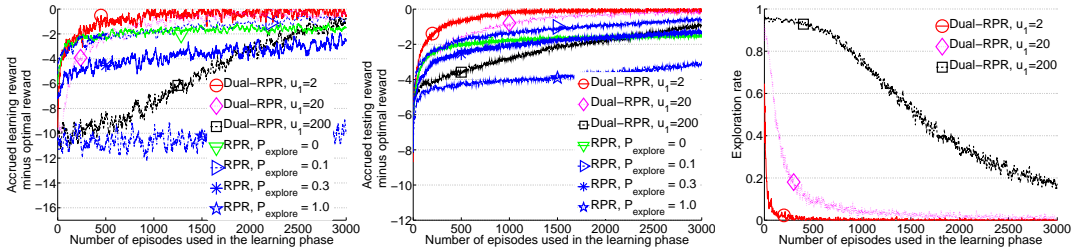

Figure 1: Results on Shuttle with a random exploration policy, with $K_{\text{total}} = 3000$. Left: accumulative discounted reward accrued within each learning episode, with the corresponding optimal reward subtracted. Middle: accumulative discounted rewards averaged over 251 episodes of following the primary RPR obtained after each learning episode, again with the corresponding optimal reward subtracted. Right: the rate of exploration in each learning episode. All results are averaged over 100 independent Monte Carlo runs.

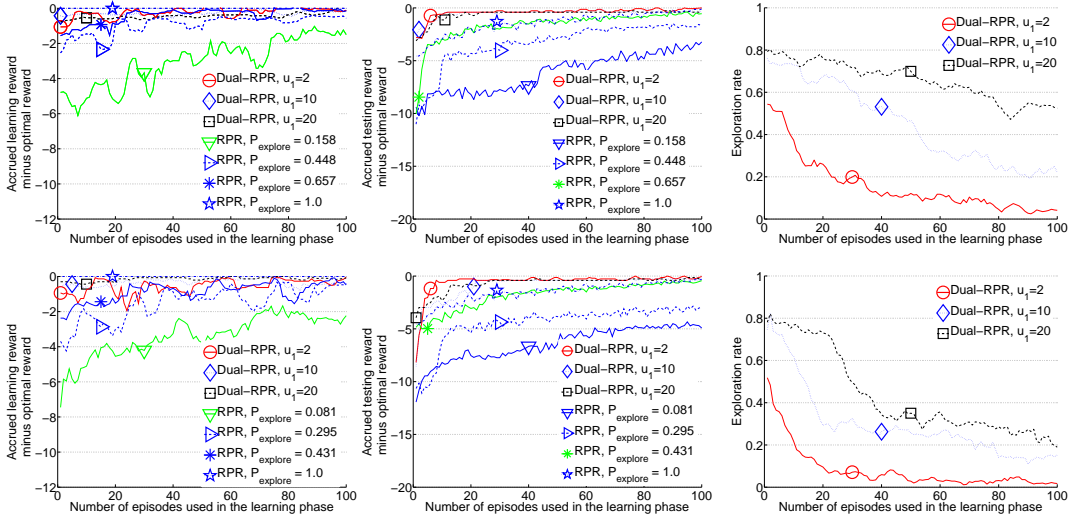

Figure 2: Results on Shuttle with an oracle exploration policy incurring cost $c = 1$ (top row) and $c = 3$ (bottom row), and $K_{\text{total}} = 100$. Each figure in a row is a counterpart of the corresponding figure in Figure 1, with the random $\Pi_e$ replaced by the oracle $\Pi_e$. See the captions there for details.

It is seen from Figure 1 that, with random exploration and $u_1 = 2$, the primary policy converges to optimality and, accordingly, $P_{\text{explore}}$ drops to zero, after about 1500 learning episodes. When $u_1$ increases to 20, the convergence is slower: it does not occur (and $P_{\text{explore}} > 0$) until after abound 2500 learning episodes. With $u_1$ increased to 200, the convergence does not happen and $P_{\text{explore}} > 0.2$ within the first 3000 learning episodes. These results verify our analysis in Section 3 and 4: (*i*) the primary policy improves as $P_{\text{explore}}$ decreases; (*ii*) the agent explores when it is not acting optimally and it is acting optimally when it stops exploring; (*iii*) there exists finite $u_1$ such that the primary policy is optimal if $P_{\text{explore}} = 0$. Although $u_1 = 2$ may still be larger than $u_1^{\min}$, it is small enough to ensure convergence within 1500 episodes. We also observe from Figure 1 that: (*i*) the agent explores more efficiently when it is adaptively switched between exploration and exploitation by the auxiliary policy, than when the switch is random; (*ii*) the primary policy cannot converge to optimality when the agent never explores; (*iii*) the primary policy may converge

to optimality when the agent always takes random actions, but it may need infinite learning episodes to converge.

The results in Figure 2, with $\Pi_e$ being an oracle, provide similar conclusions as those in Figure 1 when $\Pi_e$ is random. However, there are two special observations from Figure 2: (*i*) $P_{\text{explore}}$ is affected by the query cost $c$: with a larger $c$, the agent performs less exploration. (*ii*) the convergence rate of the primary policy is not significantly affected by the query cost. The reason for (*ii*) is that the oracle always provides optimal actions, thus over-exploration does not harm the optimality; as long as the agent takes optimal actions, the primary policy continually improves if it is not yet optimal, or it remains optimal if it is already optimal.

## 6    Conclusions

We have presented a dual-policy approach for jointly learning the agent behavior and the optimal balance between exploitation and exploration, assuming the unknown environment is a POMDP. By identifying a known part of the environment in terms of histories (parameterized by the RPR), the approach adaptively switches between exploration and exploitation depending on whether the agent is in the known part. We have provided theoretical guarantees for the agent to either explore efficiently or exploit efficiently. Experimental results show good agreement with our theoretical analysis and that our approach finds the optimal policy efficiently. Although we empirically demonstrated the existence of a small $u_1$ to ensure efficient convergence to optimality, further theoretical analysis is needed to find $u_1^{\min}$, the tight lower bound of $u_1$, which ensures convergence to optimality with just the right amount of exploration (without over-exploration). Finding the exact $u_1^{\min}$ is difficult because of the partial observability. However, it is hopeful to find a good approximation to $u_1^{\min}$. In the worst case, the agent can always choose to be optimistic, like in $E^3$ and Rmax. An optimistic agent uses a large $u_1$, which usually leads to over-exploration but ensures convergence to optimality.

## 7    Acknowledgements

The authors would like to thank the anonymous reviewers for their valuable comments and suggestions. This work is supported by AFOSR.

## Appendix

**Proof of Lemma 4.1:** We expand (10) as,

$$\widehat{V}_f(\mathcal{E}_T^{(K)}; \Theta, \lambda, \Pi_e) = \sum_{\mathcal{E}_T^{(K)}} \sum_{t=0}^{T} \gamma^t r_t p(a_{0:t}, o_{1:t}, r_t | y_{0:t} = 0, \Theta, M) p(y_{0:t} = 0 | \Theta, \lambda)$$
$$+ \sum_{\mathcal{E}_T^{(K)}} \sum_{t=0}^{T} \gamma^t R_{\max} \sum_{y_{0:t} \neq 0} p(a_{0:t}, o_{1:t}, r_t | y_{0:t}, \Theta, M, \Pi_e) p(y_{0:t} | \Theta, \lambda)$$

where $y_{0:t}$ is an an abbreviation for $y_\tau = 0 \ \forall \ \tau = 0, \cdots, t$ and $y_{0:t} \neq 0$ is an an abbreviation for $\exists \ 0 \leq \tau \leq t$ satisfying $y_\tau \neq 0$. The sum $\sum_{\mathcal{E}_T^{(K)}}$ is over all episodes in $\mathcal{E}_T^{(K)}$. The difference between (9) and (11) is

$$|\widehat{V}(\mathcal{E}_T^{(K)}, \Theta) - \widehat{V}(\mathcal{E}_T^{(K)}; \Theta, \lambda)| = \left| \sum_{\mathcal{E}_T^{(K)}} \sum_{t=0}^{T} \gamma^t r_t p(a_{0:t}, o_{1:t}, r_t | y_{0:t} = 0, \Theta, M)(1 - p(y_{0:t} = 0 | \Theta, \lambda)) \right.$$
$$\left. - \sum_{\mathcal{E}_T^{(K)}} \sum_{t=0}^{T} \gamma^t R_{\max} \sum_{y_{0:t} \neq 0} p(a_{0:t}, o_{1:t}, r_t | y_{0:t}, \Theta, M, \Pi_e) p(y_{0:t} | \Theta, \lambda) \right|$$

$$= \left| \sum_{\mathcal{E}_T^{(K)}} \sum_{t=0}^{T} \gamma^t r_t p(a_{0:t}, o_{1:t}, r_t | y_{0:t} = 0, \Theta, M) \sum_{y_{0:t} \neq 0} p(y_{0:t} | \Theta, \lambda) \right.$$
$$\left. - \sum_{\mathcal{E}_T^{(K)}} \sum_{t=0}^{T} \gamma^t R_{\max} \sum_{y_{0:t} \neq 0} p(a_{0:t}, o_{1:t}, r_t | y_{0:t}, \Theta, M, \Pi_e) p(y_{0:t} | \Theta, \lambda) \right|$$

$$= \left| \sum_{\mathcal{E}_T^{(K)}} \sum_{t=0}^{T} \gamma^t r_t \sum_{y_{0:t} \neq 0} \left[ p(a_{0:t}, o_{1:t}, r_t | y_{0:t} = 0, \Theta, M) - \frac{R_{\max}}{r_t} p(a_{0:t}, o_{1:t}, r_t | y_{0:t}, \Theta, M, \Pi_e) \right] p(y_{0:t} | \Theta, \lambda) \right|$$

$$\leq \sum_{\mathcal{E}_T^{(K)}} \sum_{t=0}^{T} \gamma^t R_{\max} \sum_{y_{0:t} \neq 0} p(y_{0:t} | \Theta, \lambda) \ = \ \sum_{\mathcal{E}_T^{(K)}} \sum_{t=0}^{T} \gamma^t R_{\max}(1 - p(y_{0:t} = 0 | \Theta, \lambda))$$

$$\leq \sum_{\mathcal{E}_T^{(K)}} (1 - p(y_{0:T} = 0 | \Theta, \lambda)) \sum_{t=0}^{T} \gamma^t R_{\max} \ \leq \ \frac{R_{\max}}{1 - \gamma} \sum_{\mathcal{E}_T^{(K)}} (1 - p(y_{0:T} = 0 | \Theta, \lambda))$$

where $\sum_{y_{0:t} \neq 0}$ is a sum over all sequences $\{y_{0:t} : \exists \ 0 \leq \tau \leq t \text{ satisfying } y_\tau \neq 0\}$.         Q.E.D.

## Footnotes

[1] An episode almost always terminates in finite time steps in practice and the agent stays in the absorbing state with zero reward for the remaining infinite steps after an episode is terminated [10]. The infinite horizon is only to ensure theoretically all episodes have the same horizon length.

# References

[1] M. J. Beal. *Variational Algorithms for Approximate Bayesian Inference*. PhD thesis, Gatsby Computational Neuroscience Unit, Univertisity College London, 2003.

[2] R. I. Brafman and M. Tennenholtz. R-max - a general polynomial time algorithm for near-optimal reinforcement learning. *Journal of Machine Learning Research*, 3(OCT):213–231, 2002.

[3] F. Doshi, J. Pineau, and N. Roy. Reinforcement learning with limited reinforcement: Using Bayes risk for active learning in POMDPs. In *Proceedings of the 25th international conference on Machine learning*, pages 256–263. ACM, 2008.

[4] M. Kearns and D. Koller. Efficient reinforcement learning in factored mdps. In *Proc. of the Sixteenth International Joint Conference of Artificial Intelligence*, pages 740–747, 1999.

[5] M. Kearns and S. P. Singh. Near-optimal performance for reinforcement learning in polynomial time. In *Proc. ICML*, pages 260–268, 1998.

[6] H. Li, X. Liao, and L. Carin. Multi-task reinforcement learning in partially observable stochastic environments. *Journal of Machine Learning Research*, 10:1131–1186, 2009.

[7] M.L. Littman, A.R. Cassandra, and L.P. Kaelbling. Learning policies for partially observable environments: scaling up. In *ICML*, 1995.

[8] J. Pineau, G. Gordon, and S. Thrun. Point-based value iteration: An anytime algorithm for POMDPs. In *Proceedings of the Sixteenth International Joint Conference on Artificial Intelligence (IJCAI)*, pages 1025 – 1032, August 2003.

[9] P. Poupart and N. Vlassis. Model-based bayesian reinforcement learning in partially observable domains. In *International Symposiu on Artificial Intelligence and Mathmatics (ISAIM)*, 2008.

[10] R. Sutton and A. Barto. *Reinforcement learning: An introduction*. MIT Press, Cambridge, MA, 1998.

